# Semi-Supervised Learning with the Graph Laplacian: The Limit of Infinite Unlabelled Data

**Boaz Nadler**
Dept. of Computer Science and Applied Mathematics
Weizmann Institute of Science
Rehovot, Israel 76100
boaz.nadler@weizmann.ac.il

**Nathan Srebro**
Toyota Technological Institute
Chicago, IL 60637
nati@uchicago.edu

**Xueyuan Zhou**
Dept. of Computer Science
University of Chicago
Chicago, IL 60637
zhouxy@cs.uchicago.edu

## Abstract

We study the behavior of the popular Laplacian Regularization method for Semi-Supervised Learning at the regime of a fixed number of labeled points but a large number of unlabeled points. We show that in $\mathbb{R}^d$, $d \geqslant 2$, the method is actually not well-posed, and as the number of unlabeled points increases the solution degenerates to a noninformative function. We also contrast the method with the Laplacian Eigenvector method, and discuss the "smoothness" assumptions associated with this alternate method.

## 1 Introduction and Setup

In this paper we consider the limit behavior of two popular semi-supervised learning (SSL) methods based on the graph Laplacian: the regularization approach [15] and the spectral approach [3]. We consider the limit when the number of labeled points is fixed and the number of unlabeled points goes to infinity. This is a natural limit for SSL as the basic SSL scenario is one in which unlabeled data is virtually infinite. We can also think of this limit as "perfect" SSL, having full knowledge of the marginal density $p(x)$. The premise of SSL is that the marginal density $p(x)$ is informative about the unknown mapping $y(x)$ we are trying to learn, e.g. since $y(x)$ is expected to be "smooth" in some sense relative to $p(x)$. Studying the infinite-unlabeled-data limit, where $p(x)$ is fully known, allows us to formulate and understand the underlying smoothness assumptions of a particular SSL method, and judge whether it is well-posed and sensible. Understanding the infinite-unlabeled-data limit is also a necessary first step to studying the convergence of the finite-labeled-data estimator.

We consider the following setup: Let $p(x)$ be an unknown smooth density on a compact domain $\Omega \subset \mathbb{R}^d$ with a smooth boundary. Let $y : \Omega \to \mathcal{Y}$ be the unknown function we wish to estimate. In case of regression $\mathcal{Y} = \mathbb{R}$ whereas in binary classification $\mathcal{Y} = \{-1, 1\}$. The standard (transductive) semi-supervised learning problem is formulated as follows: Given $l$ labeled points, $(x_1, y_1), \ldots, (x_l, y_l)$, with $y_i = y(x_i)$, and $u$ unlabeled points $x_{l+1}, \ldots, x_{l+u}$, with all points $x_i$ sampled i.i.d. from $p(x)$, the goal is to construct an estimate of $y(x_{l+i})$ for any unlabeled point $x_{l+i}$, utilizing both the labeled and the unlabeled points. We denote the total number of points by $n = l + u$. We are interested in the regime where $l$ is fixed and $u \to \infty$.

## 2 SSL with Graph Laplacian Regularization

We first consider the following graph-based approach formulated by Zhu et. al. [15]:

$$\hat{y}(x) = \arg\min_y I_n(y) \qquad \text{subject to} \qquad y(x_i) = y_i, \ i = 1, \ldots, l \qquad (1)$$

where

$$I_n(y) = \frac{1}{n^2} \sum_{i,j} W_{i,j} (y(x_i) - y(x_j))^2 \qquad (2)$$

is a Laplacian regularization term enforcing "smoothness" with respect to the $n \times n$ similarity matrix $W$. This formulation has several natural interpretations in terms of, e.g. random walks and electrical circuits [15]. These interpretations, however, refer to a fixed graph, over a finite set of points with given similarities.

In contrast, our focus here is on the more typical scenario where the points $x_i \in \mathbb{R}^d$ are a random sample from a density $p(x)$, and $W$ is constructed based on this sample. We would like to understand the behavior of the method in terms of the density $p(x)$, particularly in the limit where the number of unlabeled points grows. Under what assumptions on the target labeling $y(x)$ and on the density $p(x)$ is the method (1) sensible?

The answer, of course, depends on how the matrix $W$ is constructed. We consider the common situation where the similarities are obtained by applying some decay filter to the distances:

$$W_{i,j} = G\left(\frac{\|x_i - x_j\|}{\sigma}\right) \qquad (3)$$

where $G : \mathbb{R}^+ \to \mathbb{R}^+$ is some function with an adequately fast decay. Popular choices are the Gaussian filter $G(z) = e^{-z^2/2}$ or the $\epsilon$-neighborhood graph obtained by the step filter $G(z) = \mathbf{1}_{z<1}$.

For simplicity, we focus here on the formulation (1) where the solution is required to satisfy the constraints at the labeled points exactly. In practice, the hard labeling constraints are often replaced with a softer loss-based data term, which is balanced against the smoothness term $I_n(y)$, e.g. [14, 6]. Our analysis and conclusions apply to such variants as well.

**Limit of the Laplacian Regularization Term**

As the number of unlabeled examples grows the regularization term (2) converges to its expectation, where the summation is replaced by integration w.r.t. the density $p(x)$:

$$\lim_{n \to \infty} I_n(y) = I^{(\sigma)}(y) = \int_\Omega \int_\Omega G\left(\frac{\|x - x'\|}{\sigma}\right) (y(x) - y(x'))^2 p(x) p(x') dx dx'. \qquad (4)$$

In the above limit, the bandwidth $\sigma$ is held fixed. Typically, one would also drive the bandwidth $\sigma$ to zero as $n \to \infty$. There are two reasons for this choice. First, from a practical perspective, this makes the similarity matrix $W$ sparse so it can be stored and processed. Second, from a theoretical perspective, this leads to a clear and well defined limit of the smoothness regularization term $I_n(y)$, at least when $\sigma \to 0$ slowly enough[1], namely when $\sigma = \omega(\sqrt[d]{\log n / n})$. If $\sigma \to 0$ as $n \to \infty$, and as long as $n\sigma^d / \log n \to \infty$, then after appropriate normalization, the regularizer converges to a density weighted gradient penalty term [7, 8]:

$$\lim_{n \to \infty} \frac{d}{C\sigma^{d+2}} I_n(y) = \lim_{\sigma \to 0} \frac{d}{C\sigma^{d+2}} I^{(\sigma)}(y) = J(y) = \int_\Omega \|\nabla y(x)\|^2 p(x)^2 dx \qquad (5)$$

where $C = \int_{\mathbb{R}^d} \|z\|^2 G(\|z\|) dz$, and assuming $0 < C < \infty$ (which is the case for both the Gaussian and the step filters). This energy functional $J(f)$ therefore encodes the notion of "smoothness" with respect to $p(x)$ that is the basis of the SSL formulation (1) with the graph constructions specified by (3). To understand the behavior and appropriateness of (1) we must understand this functional and the associated limit problem:

$$\hat{y}(x) = \arg\min_y J(y) \qquad \text{subject to} \qquad y(x_i) = y_i, \ i = 1, \ldots, l \qquad (6)$$

# 3 Graph Laplacian Regularization in $\mathbb{R}^1$

We begin by considering the solution of (6) for one dimensional data, i.e. $d = 1$ and $x \in \mathbb{R}$. We first consider the situation where the support of $p(x)$ is a continuous interval $\Omega = [a, b] \subset \mathbb{R}$ ($a$ and/or $b$ may be infinite). Without loss of generality, we assume the labeled data is sorted in increasing order $a \leqslant x_1 < x_2 < \cdots < x_l \leqslant b$. Applying the theory of variational calculus, the solution $\hat{y}(x)$ satisfies inside each interval $(x_i, x_{i+1})$ the Euler-Lagrange equation

$$\frac{d}{dx}\left[p^2(x)\frac{dy}{dx}\right] = 0.$$

Performing two integrations and enforcing the constraints at the labeled points yields

$$y(x) = y_i + \frac{\int_{x_i}^x 1/p^2(t)dt}{\int_{x_i}^{x_{i+1}} 1/p^2(t)dt}(y_{i+1} - y_i) \qquad \text{for } x_i \leqslant x \leqslant x_{i+1} \tag{7}$$

with $y(x) = x_1$ for $a \leqslant x \leqslant x_1$ and $y(x) = x_l$ for $x_l \leqslant x \leqslant b$. If the support of $p(x)$ is a union of disjoint intervals, the above analysis and the form of the solution applies in each interval separately.

The solution (7) seems reasonable and desirable from the point of view of the "smoothness" assumptions: when $p(x)$ is uniform, the solution interpolates linearly between labeled data points, whereas across low-density regions, where $p(x)$ is close to zero, $y(x)$ can change abruptly. Furthermore, the regularizer $J(y)$ can be interpreted as a Reproducing Kernel Hilbert Space (RKHS) squared semi-norm, giving us additional insight into this choice of regularizer:

**Theorem 1.** *Let $p(x)$ be a smooth density on $\Omega = [a, b] \subset \mathbb{R}$ such that $A_p = \frac{1}{4}\int_a^b 1/p^2(t)dt < \infty$. Then, $J(f)$ can be written as a squared semi-norm $J(f) = \|f\|_{K_p}^2$ induced by the kernel*

$$K_p(x, x') = A_p - \frac{1}{2}\left|\int_x^{x'} \frac{1}{p^2(t)} dt\right|. \tag{8}$$

*with a null-space of all constant functions. That is, $\|f\|_{K_p}$ is the norm of the projection of $f$ onto the RKHS induced by $K_p$.*

*If $p(x)$ is supported on several disjoint intervals, $\Omega = \cup_i[a_i, b_i]$, then $J(f)$ can be written as a squared semi-norm induced by the kernel*

$$K_p(x, x') = \begin{cases} \frac{1}{4}\int_{a_i}^{b_i} \frac{dt}{p^2(t)} - \frac{1}{2}\left|\int_x^{x'} \frac{dt}{p^2(t)}\right| & \text{if } x, x' \in [a_i, b_i] \\ 0 & \text{if } x \in [a_i, b_i], x' \in [a_j, b_j], i \neq j \end{cases} \tag{9}$$

*with a null-space spanned by indicator functions $\mathbf{1}_{[a_i, b_i]}(x)$ on the connected components of $\Omega$.*

*Proof.* For any $f(x) = \sum_i \alpha_i K_p(x, x_i)$ in the RKHS induced by $K_p$:

$$J(f) = \int\left(\frac{df}{dx}\right)^2 p^2(x)dx = \sum_{i,j}\alpha_i\alpha_j J_{ij} \tag{10}$$

$$\text{where} \quad J_{ij} = \int\frac{d}{dx}K_p(x, x_i)\frac{d}{dx}K_p(x, x_j)p^2(x)dx$$

When $x_i$ and $x_j$ are in different connected components of $\Omega$, the gradients of $K_p(\cdot, x_i)$ and $K_p(\cdot, x_j)$ are never non-zero together and $J_{ij} = 0 = K_p(x_i, x_j)$. When they are in the same connected component $[a, b]$, and assuming w.l.o.g. $a \leqslant x_i \leqslant x_j \leqslant b$:

$$J_{ij} = \frac{1}{4}\left[\int_a^{x_i} \frac{1}{p^2(t)}dt + \int_{x_i}^{x_j} \frac{-1}{p^2(t)}dt + \int_{x_j}^b \frac{1}{p^2(t)}dt\right]$$

$$= \frac{1}{4}\int_a^b \frac{1}{p^2(t)}dt - \frac{1}{2}\int_{x_i}^{x_j} \frac{1}{p^2(t)}dt = K_p(x_i, x_j). \tag{11}$$

Substituting $J_{ij} = K_p(x_i, x_j)$ into (10) yields $J(f) = \sum \alpha_i\alpha_j K_p(x_i, x_j) = \|f\|_{K_p}$. $\qquad\square$

Combining Theorem 1 with the Representer Theorem [13] establishes that the solution of (6) (or of any variant where the hard constraints are replaced by a data term) is of the form:

$$y(x) = \sum_{j=1}^{l} \alpha_j K_p(x, x_j) + \sum_i \beta_i \mathbf{1}_{[a_i, b_i]}(x),$$

where $i$ ranges over the connected components $[a_i, b_i]$ of $\Omega$, and we have:

$$J(y) = \sum_{i,j=1}^{l} \alpha_i \alpha_j K_p(x_i, x_j). \tag{12}$$

Viewing the regularizer as $\|y\|_{K_p}^2$ suggests understanding (6), and so also its empirical approximation (1), by interpreting $K_p(x, x')$ as a density-based "similarity measure" between $x$ and $x'$. This similarity measure indeed seems sensible: for a uniform density it is simply linearly decreasing as a function of the distance. When the density is non-uniform, two points are relatively similar only if they are connected by a region in which $1/p^2(x)$ is low, i.e. the density is high, but are much less "similar", i.e. related to each other, when connected by a low-density region. Furthermore, there is no dependence between points in disjoint components separated by zero density regions.

## 4  Graph Laplacian Regularization in Higher Dimensions

The analysis of the previous section seems promising, at it shows that in one dimension, the SSL method (1) is well posed and converges to a sensible limit. Regretfully, in higher dimensions this is not the case anymore. In the following theorem we show that the infimum of the limit problem (6) is zero and can be obtained by a sequence of functions which are certainly not a sensible extrapolation of the labeled points.

**Theorem 2.** *Let $p(x)$ be a smooth density over $\mathbb{R}^d$, $d \geqslant 2$, bounded from above by some constant $p_{max}$, and let $(x_1, y_1), \ldots, (x_l, y_l)$ be any (non-repeating) set of labeled examples. There exist continuous functions $y_\epsilon(x)$, for any $\epsilon > 0$, all satisfying the constraints $y_\epsilon(x_j) = y_j, j = 1, \ldots, l$, such that $J(y_\epsilon) \xrightarrow{\epsilon \to 0} 0$ but $y_\epsilon(x) \xrightarrow{\epsilon \to 0} 0$ for all $x \neq x_j, j = 1, \ldots, l$.*

*Proof.* We present a detailed proof for the case of $l = 2$ labeled points. The generalization of the proof to more labeled points is straightforward. Furthermore, without loss of generality, we assume the first labeled point is at $x_0 = 0$ with $y(x_0) = 0$ and the second labeled point is at $x_1$ with $\|x_1\| = 1$ and $y(x_1) = 1$. In addition, we assume that the ball $B_1(0)$ of radius one centered around the origin is contained in $\Omega = \{x \in \mathbb{R}^d \,|\, p(x) > 0\}$.

We first consider the case $d > 2$. Here, for any $\epsilon > 0$, consider the function

$$y_\epsilon(x) = \min\left(\frac{\|x\|}{\epsilon}, 1\right)$$

which indeed satisfies the two constraints $y_\epsilon(x_i) = y_i$, $i = 0, 1$. Then,

$$J(y_\epsilon) = \int_{B_\epsilon(0)} \frac{p^2(x)}{\epsilon^2} dx \leqslant \frac{p_{\max}}{\epsilon^2} \int_{B_\epsilon(0)} dx = p_{\max}^2 V_d \, \epsilon^{d-2} \tag{13}$$

where $V_d$ is the volume of a unit ball in $\mathbb{R}^d$. Hence, the sequence of functions $y_\epsilon(x)$ satisfy the constraints, but for $d > 2$, $\inf_\epsilon J(y_\epsilon) = 0$.

For $d = 2$, a more extreme example is necessary: consider the functions

$$y_\epsilon(x) = \log\left(\frac{\|x\|^2 + \epsilon}{\epsilon}\right) \Big/ \log\left(\frac{1+\epsilon}{\epsilon}\right) \qquad \text{for } \|x\| \leqslant 1$$

and $y_\epsilon(x) = 1$ for $\|x\| > 1$. These functions satisfy the two constraints $y_\epsilon(x_i) = y_i$, $i = 0, 1$ and:

$$J(y_\epsilon) = \frac{4}{\left[\log\left(\frac{1+\epsilon}{\epsilon}\right)\right]^2} \int_{B_1(0)} \frac{\|x\|^2}{(\|x\|^2 + \epsilon)^2} p^2(x) dx \leqslant \frac{4p_{max}^2}{\left[\log\left(\frac{1+\epsilon}{\epsilon}\right)\right]^2} \int_0^1 \frac{r^2}{(r^2 + \epsilon)^2} 2\pi r \, dr$$

$$\leqslant \frac{4\pi p_{max}^2}{\left[\log\left(\frac{1+\epsilon}{\epsilon}\right)\right]^2} \log\left(\frac{1+\epsilon}{\epsilon}\right) = \frac{4\pi p_{max}^2}{\log\left(\frac{1+\epsilon}{\epsilon}\right)} \xrightarrow{\epsilon \to 0} 0. \qquad \square$$

The implication of Theorem 2 is that regardless of the values at the labeled points, as $u \to \infty$, the solution of (1) is not well posed. Asymptotically, the solution has the form of an almost everywhere constant function, with highly localized spikes near the labeled points, and so no learning is performed. In particular, an interpretation in terms of a density-based kernel $K_p$, as in the one-dimensional case, is not possible.

Our analysis also carries over to a formulation where a loss-based data term replaces the hard label constraints, as in

$$\hat{y} = \arg\min_{y(x)} \frac{1}{l} \sum_{j=1}^{l} (y(x_j) - y_j)^2 + \gamma I_n(y)$$

In the limit of infinite unlabeled data, functions of the form $y_\epsilon(x)$ above have a zero data penalty term (since they exactly match the labels) and also drive the regularization term $J(y)$ to zero. Hence, it is possible to drive the entire objective functional (the data term plus the regularization term) to zero with functions that do not generalize at all to unlabeled points.

## 4.1 Numerical Example

We illustrate the phenomenon detailed by Theorem 2 with a simple example. Consider a density $p(x)$ in $\mathbb{R}^2$, which is a mixture of two unit variance spherical Gaussians, one per class, centered at the origin and at $(4, 0)$. We sample a total of $n = 3000$ points, and label two points from each of the two components (four total). We then construct a similarity matrix using a Gaussian filter with $\sigma = 0.4$.

Figure 1 depicts the predictor $\hat{y}(x)$ obtained from (1). In fact, two different predictors are shown, obtained by different numerical methods for solving (1). Both methods are based on the observation that the solution $\hat{y}(x)$ of (1) satisfies:

$$\hat{y}(x_i) = \sum_{j=1}^{n} W_{ij} \hat{y}(x_j) / \sum_{j=1}^{n} W_{ij} \quad \text{on all unlabeled points } i = l+1, \ldots, l+u. \qquad (14)$$

Combined with the constraints of (1), we obtain a system of linear equations that can be solved by Gaussian elimination (here invoked through MATLAB's backslash operator). This is the method used in the top panels of Figure 1. Alternatively, (14) can be viewed as an update equation for $\hat{y}(x_i)$, which can be solved via the power method, or *label propagation* [2, 6]: start with zero labels on the unlabeled points and iterate (14), while keeping the known labels on $x_1, \ldots, x_l$. This is the method used in the bottom panels of Figure 1.

As predicted, $\hat{y}(x)$ is almost constant for almost all unlabeled points. Although all values are very close to zero, thresholding at the "right" threshold does actually produce sensible results in terms of the true -1/+1 labels. However, beyond being inappropriate for regression, a very flat predictor is still problematic even from a classification perspective. First, it is not possible to obtain a meaningful confidence measure for particular labels. Second, especially if the size of each class is not known a-priori, setting the threshold between the positive and negative classes is problematic. In our example, setting the threshold to zero yields a generalization error of 45%.

The differences between the two numerical methods for solving (1) also point out to another problem with the ill-posedness of the limit problem: the solution is numerically very un-stable.

A more quantitative evaluation, that also validates that the effect in Figure 1 is not a result of choosing a "wrong" bandwidth $\sigma$, is given in Figure 2. We again simulated data from a mixture of two Gaussians, one Gaussian per class, this time in 20 dimensions, with one labeled point per class, and an increasing number of unlabeled points. In Figure 2 we plot the squared error, and the classification error of the resulting predictor $\hat{y}(x)$. We plot the classification error both when a threshold of zero is used (i.e. the class is determined by $\text{sign}(\hat{y}(x))$) and with the ideal threshold minimizing the test error. For each unlabeled sample size, we choose the bandwidth $\sigma$ yielding the best test performance (this is a "cheating" approach which provides a lower bound on the error of the best method for selecting the bandwidth). As the number of unlabeled examples increases the squared error approaches 1, indicating a flat predictor. Using a threshold of zero leads to an increase in the classification error, possibly due to numerical instability. Interestingly, although the predictors become very flat, the classification error using the ideal threshold actually improves slightly. Note that

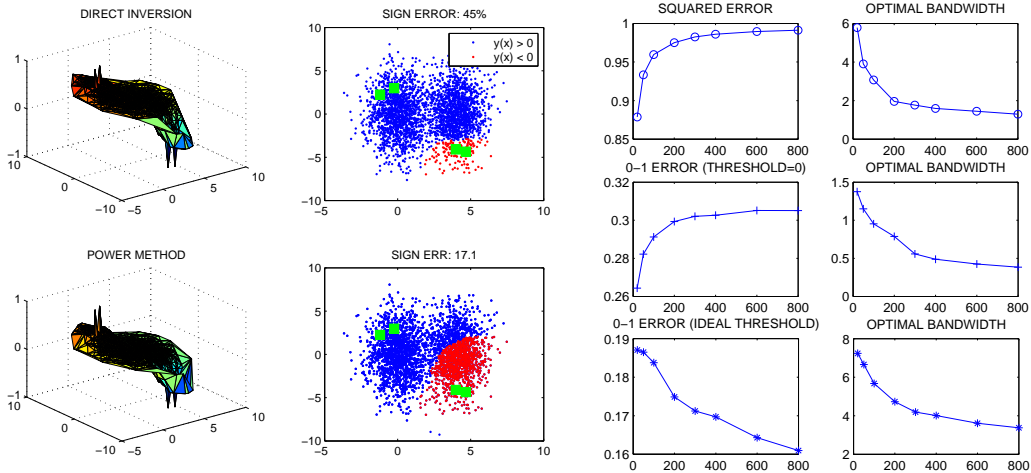

Figure 1: Left plots: Minimizer of Eq. (1). Right plots: the resulting classification according to $sign(y)$. The four labeled points are shown by green squares. Top: minimization via Gaussian elimination (MATLAB backslash). Bottom: minimization via label propagation with 1000 iterations - the solution has not yet converged, despite small residuals of the order of $2 \cdot 10^{-4}$.

Figure 2: Squared error (top), classification error with a threshold of zero (center) and minimal classification error using ideal threhold (bottom), of the minimizer of (1) as a function of number of unlabeled points. For each error measure and sample size, the bandwidth minimizing the test error was used, and is plotted.

ideal classification performance is achieved with a significantly larger bandwidth than the bandwidth minimizing the squared loss, i.e. when the predictor is even flatter.

## 4.2 Probabilistic Interpretation, Exit and Hitting Times

As mentioned above, the Laplacian regularization method (1) has a probabilistic interpretation in terms of a random walk on the weighted graph. Let $x(t)$ denote a random walk on the graph with transition matrix $M = D^{-1}W$ where $D$ is a diagonal matrix with $D_{ii} = \sum_j W_{ij}$. Then, for the binary classification case with $y_i = \pm 1$ we have [15]:

$$\hat{y}(x_i) = 2 \Pr\Big[x(t) \text{ hits a point labeled } +1 \text{ before hitting a point labeled } -1 \Big| x(0) = x_i\Big] - 1$$

We present an interpretation of our analysis in terms of the limiting properties of this random walk. Consider, for simplicity, the case where the two classes are separated by a low density region. Then, the random walk has two intrinsic quantities of interest. The first is the mean exit time from one cluster to the other, and the other is the mean hitting time to the labeled points in that cluster. As the number of unlabeled points increases and $\sigma \to 0$, the random walk converges to a diffusion process [12]. While the mean exit time then converges to a finite value corresponding to its diffusion analogue, the hitting time to a labeled point increases to infinity (as these become absorbing boundaries of measure zero). With more and more unlabeled data the random walk will fully mix, forgetting where it started, before it hits any label. Thus, the probability of hitting $+1$ before $-1$ will become uniform across the entire graph, independent of the starting location $x_i$, yielding a flat predictor.

## 5 Keeping $\sigma$ Finite

At this point, a reader may ask whether the problems found in higher dimensions are due to taking the limit $\sigma \to 0$. One possible objection is that there is an intrinsic characteristic scale for the data $\sigma_0$ where (with high probability) all points at a distance $\|x_i - x_j\| < \sigma_0$ have the same label. If this is the case, then it may not necessarily make sense to take values of $\sigma < \sigma_0$ in constructing $W$.

However, keeping $\sigma$ finite while taking the number of unlabeled points to infinity does not resolve the problem. On the contrary, even the one-dimensional case becomes ill-posed in this case. To see this, consider a function $y(x)$ which is zero everywhere except at the labeled points, where $y(x_j) = y_j$. With a finite number of labeled points of measure zero, $I^{(\sigma)}(y) = 0$ in any dimension

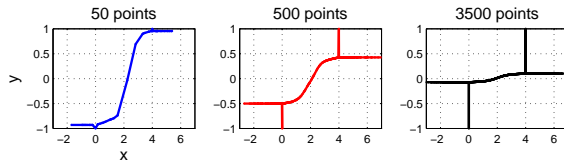

Figure 3: Minimizer of (1) for a 1-d problem with a fixed $\sigma = 0.4$, two labeled points and an increasing number of unlabeled points.

and for any fixed $\sigma > 0$. While this limiting function is discontinuous, it is also possible to construct a sequence of continuous functions $y_\epsilon$ that all satisfy the constraints and for which $I^{(\sigma)}(y_\epsilon) \xrightarrow{\epsilon \to 0} 0$.

This behavior is illustrated in Figure 3. We generated data from a mixture of two 1-D Gaussians centered at the origin and at $x = 4$, with one Gaussian labeled $-1$ and the other $+1$. We used two labeled points at the centers of the Gaussians and an increasing number of randomly drawn unlabeled points. As predicted, with a fixed $\sigma$, although the solution is reasonable when the number of unlabeled points is small, it becomes flatter, with sharp spikes on the labeled points, as $u \to \infty$.

# 6    Fourier-Eigenvector Based Methods

Before we conclude, we discuss a different approach for SSL, also based on the Graph Laplacian, suggested by Belkin and Niyogi [3]. Instead of using the Laplacian as a regularizer, constraining candidate predictors $y(x)$ non-parametrically to those with small $I_n(y)$ values, here the predictors are constrained to the low-dimensional space spanned by the first few eigenvectors of the Laplacian: The similarity matrix $W$ is computed as before, and the Graph Laplacian matrix $L = D - W$ is considered (recall $D$ is a diagonal matrix with $D_{ii} = \sum_j W_{ij}$). Only predictors

$$\hat{y}(x) = \sum_{j=1}^{p} a_j \mathbf{e}_j \tag{15}$$

spanned by the first $p$ eigenvectors $\mathbf{e}_1, \ldots, \mathbf{e}_p$ of $L$ (with smallest eigenvalues) are considered. The coefficients $a_j$ are chosen by minimizing a loss function on the labeled data, e.g. the squared loss:

$$(\hat{a}_1, \ldots, \hat{a}_p) = \arg\min \sum_{j=1}^{l} (y_j - \hat{y}(x_j))^2. \tag{16}$$

Unlike the Laplacian Regularization method (1), the Laplacian Eigenvector method (15)–(16) is well posed in the limit $u \to \infty$. This follows directly from the convergence of the eigenvectors of the graph Laplacian to the eigenfunctions of the corresponding Laplace-Beltrami operator [10, 4].

Eigenvector based methods were shown empirically to provide competitive generalization performance on a variety of simulated and real world problems. Belkin and Niyogi [3] motivate the approach by arguing that 'the eigenfunctions of the Laplace-Beltrami operator provide a natural basis for functions on the manifold and the desired classification function can be expressed in such a basis'. In our view, the success of the method is actually not due to data lying on a low-dimensional manifold, but rather due to the *low density separation* assumption, which states that different class labels form high-density clusters separated by low density regions. Indeed, under this assumption and with sufficient separation between the clusters, the eigenfunctions of the graph Laplace-Beltrami operator are approximately piecewise constant in each of the clusters, as in spectral clustering [12, 11], providing a basis for a labeling that is constant within clusters but variable across clusters. In other settings, such as data uniformly distributed on a manifold but without any significant cluster structure, the success of eigenvector based methods critically depends on how well can the unknown classification function be approximated by a truncated expansion with relatively few eigenvectors.

We illustrate this issue with the following three-dimensional example: Let $p(x)$ denote the uniform density in the box $[0,1] \times [0,0.8] \times [0,0.6]$, where the box lengths are different to prevent eigenvalue multiplicity. Consider learning three different functions, $y_1(x) = \mathbf{1}_{x_1 > 0.5}$, $y_2(x) = \mathbf{1}_{x_1 > x_2/0.8}$ and $y_3(x) = \mathbf{1}_{x_2/0.8 > x_3/0.6}$. Even though all three functions are relatively simple, all having a linear separating boundary between the classes on the manifold, as shown in the experiment described in Figure 4, the Eigenvector based method (15)–(16) gives markedly different generalization performances on the three targets. This happens both when the number of eigenvectors $p$ is set to $p = l/5$ as suggested by Belkin and Niyogi, as well as for the optimal (oracle) value of $p$ selected on the test set (i.e. a "cheating" choice representing an upper bound on the generalization error of this method).

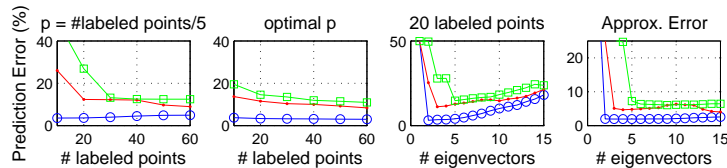

Figure 4: Left three panels: Generalization Performance of the Eigenvector Method (15)–(16) for the three different functions described in the text. All panels use $n = 3000$ points. Prediction counts the number of sign agreements with the true labels. Rightmost panel: best fit when many (all 3000) points are used, representing the best we can hope for with a few leading eigenvectors.

The reason for this behavior is that $y_2(x)$ and even more so $y_3(x)$ cannot be as easily approximated by the very few leading eigenfunctions—even though they seem "simple" and "smooth", they are significantly more complicated than $y_1(x)$ in terms of measure of simplicity implied by the Eigenvector Method. Since the density is uniform, the graph Laplacian converges to the standard Laplacian and its eigenfunctions have the form $\psi_{i,j,k}(x) = \cos(i\pi x_1)\cos(j\pi x_2/0.8)\cos(k\pi x_3/0.6)$, making it hard to represent simple decision boundaries which are not axis-aligned.

## 7   Discussion

Our results show that a popular SSL method, the Laplacian Regularization method (1), is not well-behaved in the limit of infinite unlabeled data, despite its empirical success in various SSL tasks. The empirical success might be due to two reasons.

First, it is possible that with a large enough number of labeled points relative to the number of unlabeled points, the method is well behaved. This regime, where the number of both labeled and unlabeled points grow while $l/u$ is fixed, has recently been analyzed by Wasserman and Lafferty [9]. However, we do not find this regime particularly satisfying as we would expect that having more unlabeled data available should improve performance, rather than require more labeled points or make the problem ill-posed. It also places the user in a delicate situation of choosing the "just right" number of unlabeled points without any theoretical guidance.

Second, in our experiments we noticed that although the predictor $\hat{y}(x)$ becomes extremely flat, in binary tasks, it is still typically possible to find a threshold leading to a good classification performance. We do not know of any theoretical explanation for such behavior, nor how to characterize it. Obtaining such an explanation would be very interesting, and in a sense crucial to the theoretical foundation of the Laplacian Regularization method. On a very practical level, such a theoretical understanding might allow us to correct the method so as to avoid the numerical instability associated with flat predictors, and perhaps also make it appropriate for regression.

The reason that the Laplacian regularizer (1) is ill-posed in the limit is that the first order gradient is not a sufficient penalty in high dimensions. This fact is well known in spline theory, where the Sobolev Embedding Theorem [1] indicates one must control at least $\frac{d+1}{2}$ derivatives in $\mathbb{R}^d$. In the context of Laplacian regularization, this can be done using the iterated Laplacian: replacing the graph Laplacian matrix $L = D - W$, where $D$ is the diagonal degree matrix, with $L^{\frac{d+1}{2}}$ (matrix to the $\frac{d+1}{2}$ power). In the infinite unlabeled data limit, this corresponds to regularizing all order-$\frac{d+1}{2}$ (mixed) partial derivatives. In the typical case of a low-dimensional manifold in a high dimensional ambient space, the order of iteration should correspond to the intrinsic, rather then ambient, dimensionality, which poses a practical problem of estimating this usually unknown dimensionality. We are not aware of much practical work using the iterated Laplacian, nor a good understanding of its appropriateness for SSL.

A different approach leading to a well-posed solution is to include also an ambient regularization term [5]. However, the properties of the solution and in particular its relation to various assumptions about the "smoothness" of $y(x)$ relative to $p(x)$ remain unclear.

**Acknowledgments**   The authors would like to thank the anonymous referees for valuable suggestions. The research of BN was supported by the Israel Science Foundation (grant 432/06).

## Footnotes

[1] When $\sigma = o(\sqrt[d]{1/n})$ then all non-diagonal weights $W_{i,j}$ vanish (points no longer have any "close by" neighbors). We are not aware of an analysis covering the regime where $\sigma$ decays roughly as $\sqrt[d]{1/n}$, but would be surprised if a qualitatively different meaningful limit is reached.

# References

[1] R.A. Adams, Sobolev Spaces, Academic Press (New York), 1975.

[2] A. Azran, The rendevous algorithm: multiclass semi-supervised learning with Markov Random Walks, ICML, 2007.

[3] M. Belkin, P. Niyogi, Using manifold structure for partially labelled classification, NIPS, vol. 15, 2003.

[4] M. Belkin and P. Niyogi, Convergence of Laplacian Eigenmaps, NIPS 19, 2007.

[5] M. Belkin, P. Niyogi and S. Sindhwani, Manifold Regularization: A Geometric Framework for Learning from Labeled and Unlabeled Examples, JMLR, 7:2399-2434, 2006.

[6] Y. Bengio, O. Delalleau, N. Le Roux, label propagation and quadratic criterion, in Semi-Supervised Learning, Chapelle, Scholkopf and Zien, editors, MIT Press, 2006.

[7] O. Bosquet, O. Chapelle, M. Hein, Measure Based Regularization, NIPS, vol. 16, 2004.

[8] M. Hein, Uniform convergence of adaptive graph-based regularization, COLT, 2006.

[9] J. Lafferty, L. Wasserman, Statistical Analysis of Semi-Supervised Regression, NIPS, vol. 20, 2008.

[10] U. von Luxburg, M. Belkin and O. Bousquet, Consistency of spectral clustering, Annals of Statistics, vol. 36(2), 2008.

[11] M. Meila, J. Shi. A random walks view of spectral segmentation, AI and Statistics, 2001.

[12] B. Nadler, S. Lafon, I.G. Kevrekidis, R.R. Coifman, Diffusion maps, spectral clustering and eigenfunctions of Fokker-Planck operators, NIPS, vol. 18, 2006.

[13] B. Schölkopf, A. Smola, *Learning with Kernels*, MIT Press, 2002.

[14] D. Zhou, O. Bousquet, T. Navin Lal, J. Weston, B. Schölkopf, Learning with local and global consistency, NIPS, vol. 16, 2004.

[15] X. Zhu, Z. Ghahramani, J. Lafferty, Semi-Supervised Learning using Gaussian fields and harmonic functions, ICML, 2003.

